# Perceptual Organization Based on Temporal Dynamics

**Xiuwen Liu and DeLiang L. Wang**
Department of Computer and Information Science
Center for Cognitive Science
The Ohio State University, Columbus, OH 43210-1277
*Email:* {liux, dwang}@cis.ohio-state.edu

## Abstract

A figure-ground segregation network is proposed based on a novel boundary pair representation. Nodes in the network are boundary segments obtained through local grouping. Each node is excitatorily coupled with the neighboring nodes that belong to the same region, and inhibitorily coupled with the corresponding paired node. Gestalt grouping rules are incorporated by modulating connections. The status of a node represents its probability being figural and is updated according to a differential equation. The system solves the figure-ground segregation problem through temporal evolution. Different perceptual phenomena, such as modal and amodal completion, virtual contours, grouping and shape decomposition are then explained through local diffusion. The system eliminates combinatorial optimization and accounts for many psychophysical results with a fixed set of parameters.

## 1 Introduction

Perceptual organization refers to the ability of grouping similar features in sensory data. This, at a minimum, includes the operations of grouping and figure-ground segregation, which refers to the process of determining relative depths of adjacent regions in input data and thus proper occlusion hierarchy. Perceptual organization has been studied extensively and many of the existing approaches [5] [4] [8] [10] [3] start from detecting discontinuities, i.e. edges in the input; one or several configurations are then selected according to certain criteria, for example, non-accidentalness [5]. Those approaches have several disadvantages for perceptual organization. Edges should be localized between regions and an additional ambiguity, the ownership of a boundary segment, is introduced, which is equivalent to figure-ground segregation [7]. Due to that, regional attributions cannot be associated with boundary segments. Furthermore, because each boundary segment can belong to different regions, the potential search space is combinatorial.

To overcome some of the problems, we propose a laterally-coupled network based on a boundary-pair representation to resolve figure-ground segregation. An occluding boundary is represented by a pair of boundaries of the two associated regions, and

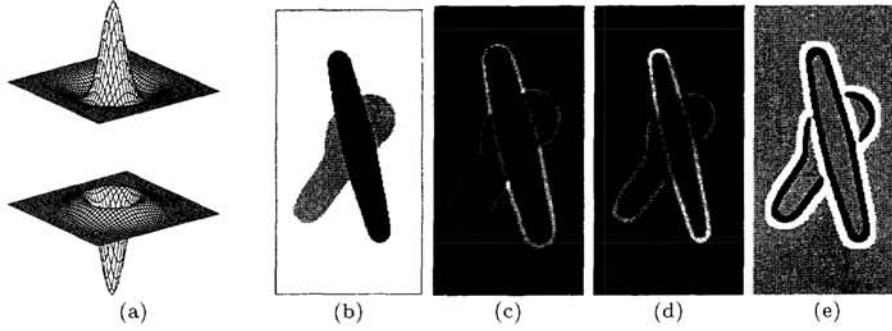

Figure 1: On- and off-center cell responses. (a) On- and off-center cells. (b) Input image. (c) On-center cell responses. (d) Off-center cell responses (e) Binarized on- and off-center cell responses, where white regions represent on-center response regions and black off-center regions.

initiates a competition between the regions. Each node in the network represents a boundary segment. Regions compete to be figural through boundary-pair competition and figure-ground segregation is resolved through temporal evolution. Gestalt grouping rules are incorporated by modulating coupling strengths between different nodes within a region, which influences the temporal dynamics and determines the percept of the system. Shape decomposition and grouping are then implemented through local diffusion using the results from figure-ground segregation.

## 2  Figure-Ground Segregation Network

The central problem in perceptual organization is to determine relative depths among regions. As figure reversal occurs in certain circumstances, figure-ground segregation cannot be resolved only based on local attributes.

### 2.1  The Network Architecture

The boundary-pair representation is motivated by on- and off-center cells, shown in Fig. 1(a). Fig. 1(b) shows an input image and Fig. 1(c) and (d) show the on- and off-center responses. Without zero-crossing, we naturally obtain double responses for each occluding boundary, as shown in Fig. 1(e). In our boundary-pair representation, each boundary is uniquely associated with a region.

In this paper, we obtain closed region boundaries from segmentation and form boundary segments using corners and junctions, which are detected through local corner and junction detectors. A node $i$ in the figure-ground segregation network represents a boundary segment, and $P_i$ represents its probability being figural, which is set to 0.5 initially. Each node is laterally coupled with neighboring nodes on the closed boundary. The connection weight from node $i$ to $j$, $w_{ij}$, is 1 and can be modified by T-junctions and local shape information. Each occluding boundary is represented by a pair of boundary segments of the involved regions. For example, in Fig. 2(a), nodes 1 and 5 form a boundary pair, where node 1 belongs to the white region and node 5 belongs to the black region. Node $i$ updates its status by:

$$\tau \frac{dP_i}{dt} = \mu_L \sum_{k \in N(i)} w_{ki}(P_k - P_i) + \mu_J(1 - P_i) \sum_{l \in J(i)} H(Q_{li}) + \mu_B(1 - P_i) \exp(-\frac{B_i}{K_B}) \quad (1)$$

Here $N(i)$ is the set of neighboring nodes of $i$, and $\mu_L$, $\mu_J$, and $\mu_B$ are parameters to determine the influences from lateral connections, junctions, and bias. $J(i)$ is

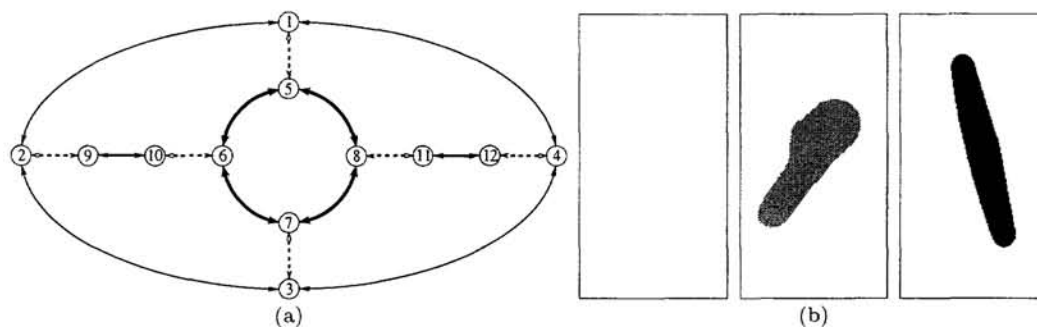

Figure 2: (a) The figure-ground segregation network for Fig. 1(b). Nodes 1, 2, 3 and 4 belong to the white region; nodes 5, 6, 7, and 8 belong to the black region; and nodes 9 and 10, and nodes 11 and 12 belong to the left and right gray regions respectively. Solid lines represent excitatory coupling while dashed lines represent inhibitory connections. (b) Result after surface completion. Left and right gray regions are grouped together.

the set of junctions that are associated with $i$ and $Q_{li}$ is the junction strength of node $i$ of junction $l$. $H(x)$ is given by $H(x) = tanh(\beta(x - \theta_J))$, where $\beta$ controls the steepness and $\theta_J$ is a threshold.

In (1), the first term on the right reflects the lateral influences. When nodes are strongly coupled, they are more likely to be in the same status, either figure or background. The second term incorporates junction information. In other words, at a T-junction, segments that vary more smoothly are more likely to be figural. The third term is a bias, where $B_i$ is the bias introduced to simulate human perception. The competition between paired nodes $i$ and $j$ is through normalization based on the assumption that only one of the paired nodes should be figural at a given time: $P_i^{(t+1)} = P_i^t/(P_i^t + P_j^t)$ and $P_j^{(t+1)} = P_j^t/(P_i^t + P_j^t)$.

## 2.2   Incorporation of Gestalt Rules

To generate behavior that is consistent with human perception, we incorporate grouping cues and some Gestalt grouping principles. As the network provides a generic model, additional grouping rules can also be incorporated.

**T-junctions**   T-junctions provide important cues for determining relative depths [7] [10].   In Williams and Hanson's model [10], T-junctions are imposed as topological constraints. Given a T-junction $l$, the initial strength for node $i$ that is associated with $l$ is:

$$Q_{li} = \frac{\exp(-\alpha_{(i,c(i))}/K_T)}{1/2 \sum_{k \in N_J(l)} \exp(-\alpha_{(k,c(k))}/K_T)},$$

where $K_T$ is a parameter, $N_J(l)$ is a set of all the nodes associated with junction $l$, $c(i)$ is the other node in $N_J(l)$ that belongs to the same region as node $i$, and $\alpha_{(ij)}$ is the angle between segments $i$ and $j$.

**Non-accidentalness**   Non-accidentalness tries to capture the intrinsic relationships among segments [5]. In our system, an additional connection is introduced to node $i$ if it is aligned well with a node $j$ from the same region and $j \notin N(i)$ initially. The connection weight $w_{ij}$ is a function of distance and angle between the involved ending points. This can be viewed as virtual junctions, resulting in virtual contours and conversion of a corner into a T-junction if involved nodes become figural. This corresponds to an organization criterion proposed by Geiger et al [3].

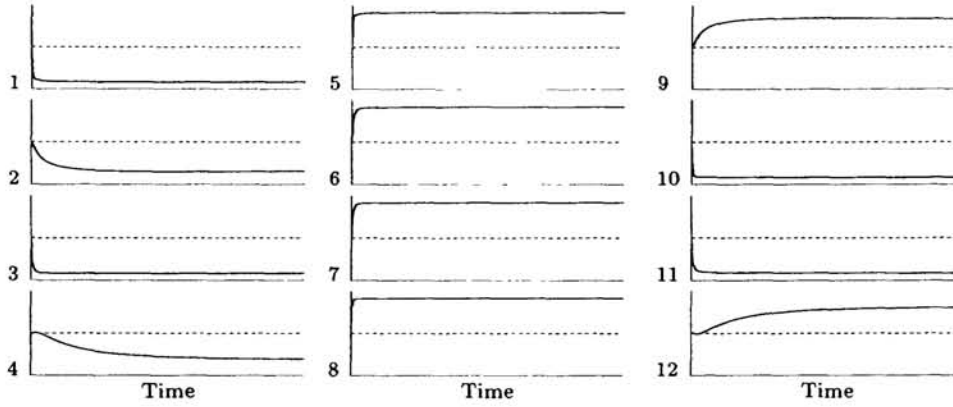

Figure 3: Temporal behavior of each node in the network shown in Fig. 2(a). Each plot shows the status of the corresponding node with respect to time. The dashed line is 0.5.

**Shape information** Shape information plays a central role in Gestalt principles and is incorporated through enhancing lateral connections. In this paper, we consider local symmetry. Let $j$ and $k$ be two neighboring nodes of $i$:

$$w_{ij} = 1 + C \exp(-|\alpha_{ij} - \alpha_{ki}|/K_\alpha) * \exp(-(L_j/L_k + L_k/L_j - 2)/K_L)),$$

where $C$, $K_\alpha$, and $K_L$ are parameters and $L_j$ is the length of segment $j$. Essentially the lateral connections are strengthened when two neighboring segments of $i$ are symmetric.

**Preferences** Human perceptual systems often prefer some organizations over others. Here we incorporated a well-known figure-ground segregation principle, called closeness. In other words, the system prefers filled regions over holes. In current implementation, we set $B_i = 1.0$ if node $i$ is part of a hole and otherwise $B_i = 0$.

### 2.3 Temporal Properties of the Network

After we construct the figure-ground segregation network, each node is updated according to (1). Fig. 3 shows the temporal behavior of the network shown in Fig. 2(a). The system approaches to a stable solution. For figure-ground segregation, we can binarize the status of each node using threshold 0.5. Thus the system generates the desired percept in a few iterations. The black region occludes other regions while gray regions occlude the white region. For example, $P_5$ is close to 1 and thus segment 5 is figural, and $P_1$ is close to 0 and thus segment 1 is in the background.

### 2.4 Surface Completion

After figure-ground segregation is resolved, surface completion and shape decomposition are implemented through diffusion [3]. Each boundary segment is associated with regional attributes such as the average intensity value because its ownership is known. Boundary segments are then grouped into diffusion groups based on similarities of their regional attributes and if they are occluded by common regions. In Fig. 1(b), three diffusion groups are formed, namely, the black region, two gray regions, and the white region. Segments in one diffusion group are diffused simultaneously. For a figural segment, a buffer with a given radius is generated. Within the buffer, the values are fixed to 1 for pixels belonging to the region and 0 otherwise. Now the problem becomes a well-defined mathematical problem. We need to solve

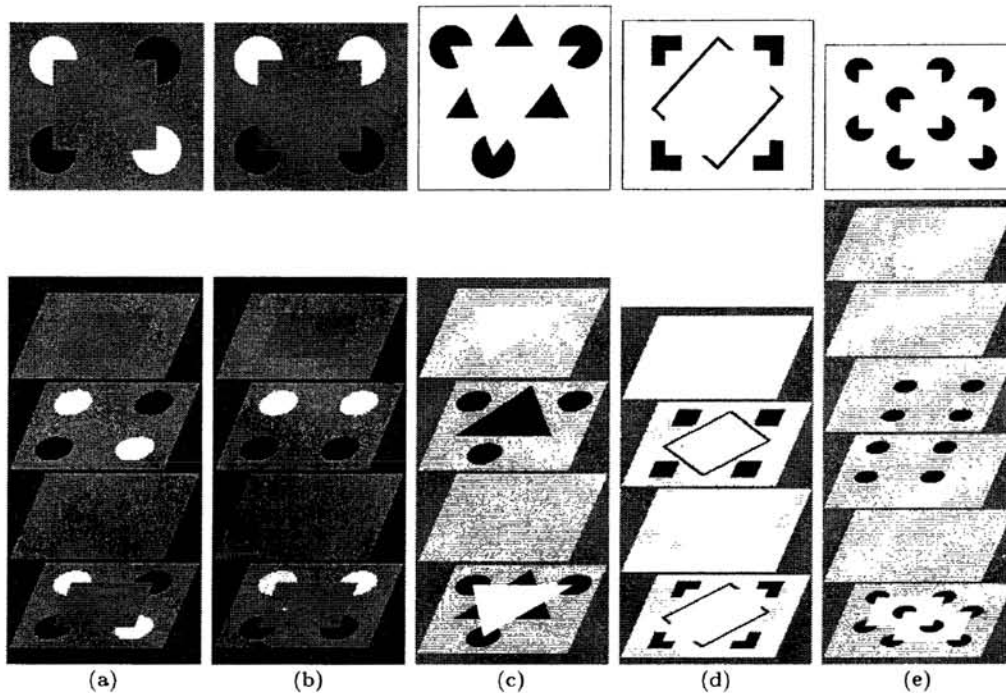

Figure 4: Images with virtual contours. In each column, the top shows the input image and the bottom the surface completion result, where completed surfaces are shown according to their relative depths and the bottom one is the projection of all the completed surfaces. (a) Alternate pacman. (b) Reverse-contrast pacman. (c) Kanizsa triangle. (d) Woven square. (e) Double pacman.

the heat equation with given boundary conditions. Currently, the heat equation is solved through local diffusion. The results from diffusion are then binarized using threshold 0.5. Fig. 2(b) shows the results for Fig. 1(b) after surface completion. Here the two gray regions are grouped together through surface completion because occluded boundaries allow diffusion. The white region becomes the background, which is the entire image.

# 3   Experimental Results

Given an image, the system automatically constructs the network and establishes the connections based on the rules discussed in Section 2.2. For all the experiments shown here, a fixed set of parameters is used.

## 3.1   Modal and Amodal Completion

We first demonstrate that the system can simulate virtual contours and modal completion. Fig. 4 shows the input images and surface completion results. The system correctly solves figure-ground segregation problem and generates the most probable percept. Fig. 4 (a) and (b) show two variations of pacman images [9] [4]. Even though the edges have opposite contrast, the virtual rectangle is vivid. Through boundary-pair representation, our system can handle both cases using the same network. Fig. 4(c) shows a typical virtual image [6] and the system correctly simulates the percept. In Fig. 4(d) [6], the rectangular-like frame is tilted, making the order between the frame and virtual square not well-defined. Our system handles that in the temporal domain. At any given time, the system outputs one of the

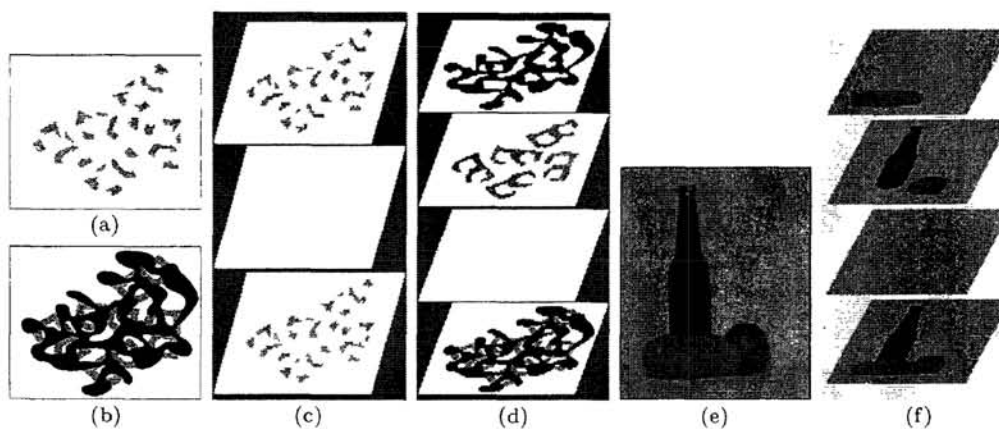

Figure 5: Surface completion results. (a) and (b) Bregman figures [1]. (c) and (d) Surface completion results for (a) and (b). (e) and (f) An image of some groceries and surface completion result.

completed surfaces. Due to this, the system can also handle the case in Fig. 4(e) [2], where the percept is bistable, as the order between the two virtual squares is not well defined.

Fig. 5(a) and (b) show the well-known Bregman figures [1]. In Fig. 5(a), there is no perceptual grouping and parts of B's remain fragmented. However, when occlusion is introduced as in Fig. 5(b), perceptual grouping is evident and fragments of B's are grouped together. Our results, shown in Fig. 5 (c) and (d), are consistent with the percepts. Fig. 5(e) shows an image of groceries, which is used extensively in [8]. Even though the T-junction at the bottom is locally confusing, our system gives the most plausible result through lateral influences of the other two strong T-junctions. Without search and parameter tuning, our system gives the optimal solution shown in Fig. 5(f).

## 3.2  Comparison with Existing Approaches

As mentioned earlier, at the minimum, figure-groud segregation and grouping need to be addresssed for perceptual organization. Edge-based approaches [4] [10] attempt to solve both problems simultaneously by preferring some configurations over combinatorially many ones according to certain creteria. There are several difficulties common to those approaches. First it cannot account for different human percepts of cases where edge elements are similar. Fig. 5 (a) and (b) are well-known examples in this regard. Another example is that the edge-only version of Fig. 4(c) does not give rise to a vivid virtual contour as in Fig. 4(c) [6]. To reduce the potential search space, often contrast signs of edges are used as additional contraints [10]. However, both Fig. 4 (a) and (b) give rise to virtual contours despite the opposite edge contrast signs. Essentially based on Fig. 4(b), Grossberg and Mingolla [4] claimed that illusory contours can join edges with different directions of contrast, which does not hold in general. As demonstrated through experiments, our approach does offer a common principle underlying these examples.

Our approach shares some similarities with the one by Geiger et al [3]. In both approaches, perceptual organization is solved in two steps. In [3], figure-ground segregation is encoded implicitly in hypotheses which are defined at junction points. Because potential hypotheses are combinatorial, only a few manually chosen ones are tested in their experiments, which is not sufficient for a general computational

model. In our approach, by resolving figure-ground segregation, there is no need to define hypotheses explicitly. In both methods, grouping is implemented through diffusion. In [3], "heat" sources for diffusion are given manually for each hypothesis whereas our approach generates "heat" sources automatically using the figure-ground segregation results. Finally, in our approach, local ambiguities can be resolved through lateral connections using temporal dynamics, resulting in robust behavior. To obtain good results for Fig. 5(e), Nitzberg et al [8] need to tune parameters and increase their search space substantially due to the misleading T-junction at the bottom of Fig. 5(e).

## 4   Conclusion

In this paper we have proposed a network for perceptual organization using temporal dynamics. The pair-wise boundary representation resolves the ownership ambiguity inherent in an edge-based representation and is equivalent to a surface representation through diffusion, providing a unified edge- and surface-based representation. Through temporal dynamics, our model allows for interactions among different modules and top-down influences can be incorporated.

## Acknowledgments

Authors would like to thank S. C. Zhu and M. Wu for their valuable discussions. This research is partially supported by an NSF grant (IRI-9423312) and an ONR Young Investigator Award (N00014-96-1-0676) to DLW.

## References

[1] A. S. Bregman, "Asking the 'What for' question in auditory perception," In *Perceptual Organization*, M. Kubovy and J R. Pomerantz, eds., Lawrence Erlbaum Associates, Publishers, Hillsdale, New Jersey, pp. 99-118, 1981.

[2] M. Fahle and G. Palm, "Perceptual rivalry between illusory and real contours," *Biological Cybernetics*, vol. 66, pp. 1-8, 1991.

[3] D. Geiger, H. Pao, and N. Rubin, "Salient and multiple illusory surfaces," In *Proceedings of IEEE Computer Society Conference on Computer Vision and Pattern Recognition*, pp. 118-124, 1998.

[4] S. Grossberg and E. Mingolla, "Neural dynamics of perceptual grouping: textures, boundaries, and emergent segmentations," *Perception & Psychophysics*, vol. 38, pp. 141-170, 1985.

[5] D. G. Lowe, *Perceptual Organization and Visual Recognition*, Kluwer Academic Publishers, Boston, 1985.

[6] G. Kanizsa, *Organization in Vision*, Praeger, New York, 1979.

[7] K. Nakayama, Z. J. He, and S. Shimojo, "Visual surface representation: a critical link between lower-level and higher-level vision," In *Visual Cognition*, S. M. Kosslyn and D. N. Osherson, eds., The MIT Press, Cambridge, Massachusetts, vol. 2, pp. 1-70, 1995.

[8] M. Nitzberg, D. Mumford, and T. Shiota, *Filtering, Segmentation and Depth*, Springer-Verlag, New York, 1993.

[9] R. Shapley and J. Gordon, "The existence of interpolated illusory contours depends on contrast and spatial separation," In *The Perception of Illusory Contours*, S. Petry and G. E. Meyer, eds., Springer-Verlag, New York, pp. 109-115, 1987.

[10] L. R. Williams and A. R. Hanson, "Perceptual Completion of Occluded Surfaces," *Computer Vision and Image Understanding*, vol. 64, pp. 1-20, 1996.